# Recursive ICA

**Honghao Shan,    Lingyun Zhang,    Garrison W. Cottrell**
Department of Computer Science and Engineering
University of California, San Diego
La Jolla, CA 92093-0404
{hshan,lingyun,gary}@cs.ucsd.edu

## Abstract

Independent Component Analysis (ICA) is a popular method for extracting independent features from visual data. However, as a fundamentally linear technique, there is always nonlinear residual redundancy that is not captured by ICA. Hence there have been many attempts to try to create a hierarchical version of ICA, but so far none of the approaches have a natural way to apply them more than once. Here we show that there is a relatively simple technique that transforms the absolute values of the outputs of a previous application of ICA into a normal distribution, to which ICA maybe applied again. This results in a recursive ICA algorithm that may be applied any number of times in order to extract higher order structure from previous layers.

## 1   Introduction

Linear implementations of Barlow's efficient encoding hypothesis[1], such as ICA [1] and sparse coding [2], have been used to explain the very first layers of auditory and visual information processing in the cerebral cortex [1, 2, 3]. Nevertheless, many interesting structures are nonlinear functions of the stimulus inputs, which are unlikely to be captured by a linear model. For example, for natural images, it has been observed that there is still significant statistical dependency between the variance of the filter outputs [4]. Several extensions of the linear ICA algorithm [5, 6, 7, 8] have been proposed to reduce such residual nonlinear redundancy, with an explicit or implicit aim of explaining higher perceptual layers, such as complex cells in V1. However, none of these extensions are obviously recursive, so it is unclear how to generalize them to multi-layer models in order to account for even higher perceptual layers.

In this paper, we propose a hierarchical redundancy reduction model in which the problem of modeling the residual nonlinear dependency is transformed into another LEE problem, as illustrated in Figure 1. There are at least two reasons why we want to do this. First, this transforms a new and hard problem into an easier and previously solved problem. Second, different parts of the brain share similar anatomical structures and it is likely that they are also working under similar computational principles. For example, fMRI studies have shown that removal of one sensory modality leads to neural reorganization of the remaining modalities [9], suggesting that the same principles must be at work across modalities. Since the LEE model has been so successful in explaining the very first layer of perceptual information processing in the cerebral cortex, it seems reasonable to hypothesize that higher layers might also be explained by a LEE model.

The problem at hand is then how to transform the problem of modeling the residual nonlinear dependency into a LEE problem. To achieve this goal, we need to first make clear what the input constraints are that are imposed by the LEE model. This is done in Section 2. After that, we will derive the transformation function that "prepares" the output of ICA for its recursive application, and then test this model on natural images.

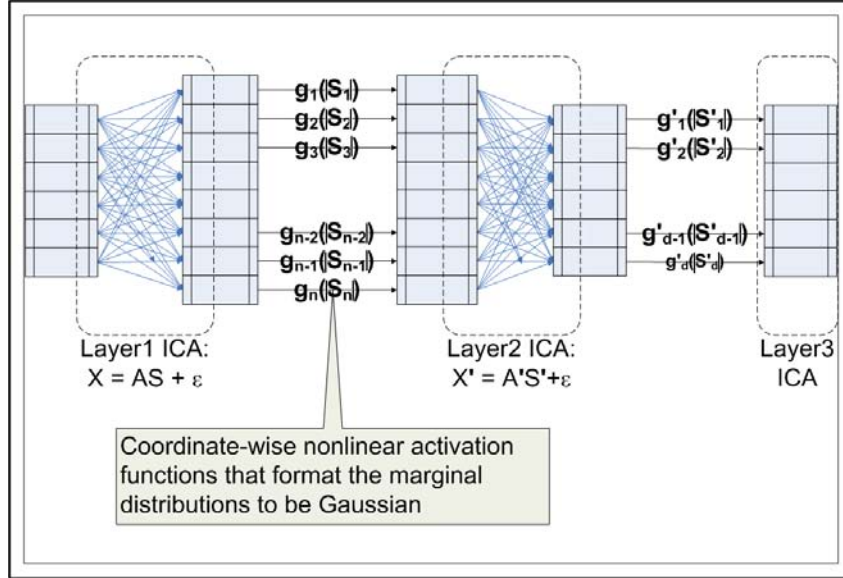

Figure 1: The RICA (Recursive ICA) model. After the first layer of linear efficient encoding, sensory inputs $X$ are now represented by $S$. The signs of $S$ are discarded. Then coordinate-wise nonlinear activation functions $g_i$ are applied to each dimension of $S$, so that the input of the next layer $X' = g(|S|)$ satisfies the input constraints imposed by the LEE model. The statistical structure among dimensions of $X'$ are then extracted by the next layer of linear efficient encoding.

## 2 Bayesian Explanation of Linear Efficient Encoding

It has long been hypothesized that the functional role of perception is to capture the statistical structure of the sensory stimuli so that appropriate action decisions can be made to maximize the chance of survival (see [10] for a brief review). Barlow provided the insight that the statistical structure is measured by the redundancy of the stimuli and that completely independent stimuli cannot be distinguished from random noise [11]. He also hypothesized that one way for the neural system to capture the statistical structure is to remove the redundancy in the sensory outputs. This so-called *redundancy reduction principle* forms the foundation of ICA algorithms.

Algorithms following the sparse coding principle are also able to find interesting structures when applied to natural image patches [2]. Later it was realized that although ICA and sparse coding algorithms started out from different principles and goals, their implementations can be summarized in the same Bayesian framework [12]. In this framework, the observed data $X$ is assumed to be generated by some underlying signal sources $S$:

$$X = AS + \epsilon$$

where $A$ is a linear mixing matrix and $\epsilon$ is additive Gaussian noise. Also, it is assumed that the features $S_j$ are independent from each other, and that the marginal distribution of $S_j$ is sparse. For the sparse coding algorithm described in [2], although it started from the goal of finding sparse features, the algorithm's implementation implicitly assumes the independence of $S_j$'s. For the infomax ICA algorithm [1], although it aimed at finding independent features, the algorithm's implementation assumes a sparse marginal prior ($p(S_j) \propto \mathrm{sech}(S_j)$). The energy-based ICA algorithm using a student-t prior [13] can also be placed in this framework for complete representations.

The moral here, though, is that in practice, the samples available are always insufficient to allow any efficient inference without making some assumptions about the data distribution. A sparseness and independence assumption about the data distribution is appropriate because: (1) independence allows the system to capture the statistical structure of the stimuli, as described above, and (2) sparse distribution of the sensory outputs is energy-economic. This is important for the survival of the biological system, considering the fact that human brain consists $2\%$ of the body weight but accounts for $20\%$ of its resting metabolism [14]. The linear efficient encoding model captures the

important characteristics of sensory coding: capturing the statistical structure (independence) of sensory stimuli with minimum cost (sparseness).

This generative model describes our assumption about the data. How well the algorithms perform depends on how well this assumption matches the real data. Hence, it is very important to check what kind of data the model generates. If the input data strongly deviate from what can be generated by the model (in other words, the observed data strongly deviate from our assumption), the results could be errant no matter how much effort we put into the model parameter estimation. As to the LEE model, there is a clear constraint on the marginal distribution of $X_i$.

Here we limit our study on those ICA algorithms that produce basis functions resembling the simple cells' receptive fields when applied on natural image patches. Such algorithms [1, 13, 15] typically adopt a symmetric [2] and sparse marginal prior for $S_j$'s that can be well approximated by a generalized Gaussian distribution. In fact, if we apply linear filters resembling the receptive fields of simple cells on natural images, the distribution of the filter responses can be well approximated by a generalized Gaussian distribution.

Here we show that such a prior suggests that the $X_i$'s should also be symmetric. A random variable $X$ is symmetric if and only if its characteristic function is real valued. In the above Bayesian framework, we assume that $S_j$'s are independent and the marginal distribution of $S_j$ is symmetric about zero. The characteristic function is then given by:

$$E[e^{\sqrt{-1}tX_i}] \quad = \quad E[e^{\sqrt{-1}t\sum_j A_{i,j}S_j}] \quad (X_i = \sum_j A_{i,j}S_j) \tag{1}$$

$$= \quad E[\prod_j e^{\sqrt{-1}tA_{i,j}S_j}] \tag{2}$$

$$= \quad \prod_j E[e^{\sqrt{-1}tA_{i,j}S_j}] \quad (S_j\text{'s are independent from each other}) \tag{3}$$

Since $A_{i,j}S_j$ is symmetric, it is easy to see that $X_i$ must also be symmetric.

A surprising fact about our perceptual system is that there does exist such a process that regularizes the marginal distribution of the sensory inputs. In the visual system, for example, the data is whitened in the retina and the LGN before transmission to V1. The functional role of this process is generally described as removing pairwise redundancy, as natural images (as well as natural sounds) obey the $1/f$ power law in the frequency domain [16]. However, as shown in Figure 2, it also regulates the marginal distribution of the input to follow a generalized-gaussian-like distribution[3].

This phenomenon has long been observed. We believe that besides the functional role of removing second-order redundancy, whitening might also serve as a role of formatting the sensory input for the cortex. For example, it has been observed [1] that without pre-whitening the images, the learned basis functions by ICA do not cover a broad range of spatial frequencies.

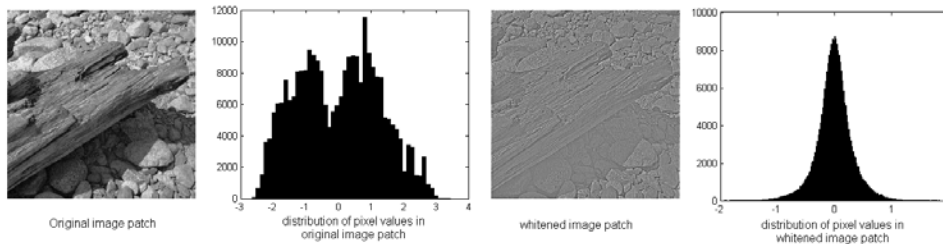

Figure 2: The distribution of pixel values of whiten images follows a generalized Gaussian distribution (see Section 2). The shape parameter of the distribution is about 1.094, which means that the marginal distribution of the inputs to the LEE model is already very sparse.

In this work, we will make the assumption that the marginal distribution of the inputs to the LEE model is a generalized gaussian distribution, as this enables the LEE model to work more efficiently. Also, as just discussed, at least for sound and image processing, there is an effective way to achieve this neurally.

# 3 Reducing Residual Redundancy

For the filter outputs $S$ of a layer of LEE, we will first discard information that provides no interesting structure (i.e., redundancy), and find an activation function such that the marginal distribution obeys the input requirements of the next layer.

## 3.1 Discarding the Signs

It has been argued that the signs of the filter outputs do not carry any redundancy [5]. The models proposed in [6, 7, 8] also implicitly or explicitly discard the signs. We have observed the usefulness of this process in a study of natural image statistics. We applied the FastICA algorithm [15] to 20x20 natural image patches, and studied the joint distribution of the filter outputs. As shown in the left plot of Figure 3, $p(s_i|s_j) = p(s_i|-s_j)$, i.e. the conditional probability of $s_i$ on $s_j$ only depends on the absolute value of $s_j$. In other words, the signs of $S$ do not provide any dependency among the dimensions. By removing the sign and applying our transformation (described in the next section), the nonlinear dependency between the $s_i$'s is exposed (see Figure 3, right).

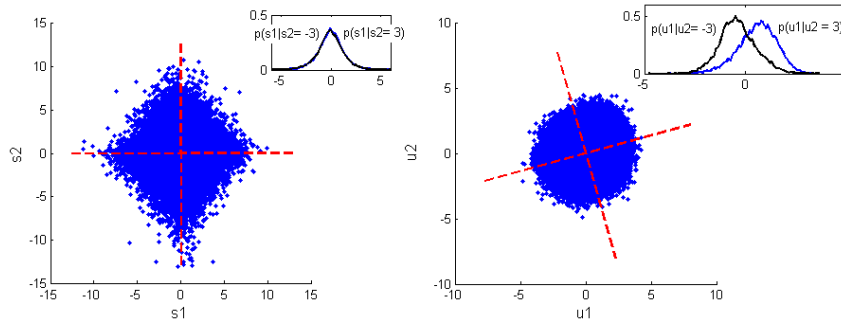

Figure 3: Left: $s_1$ and $s_2$ are ICA filter responses on natural image patches. The red dashed lines plot the linear regression between them. Right: After the coordinate-wise nonlinear transformation, the two features are no longer uncorrelated.

## 3.2 Nonlinear Activation Function

The only problem left is to find the coordinate-wise activation function $g_i$ for each dimension of $S$ such that $X'_i = g_i(|S_i|)$ follows a generalized Gaussian distribution, as required by the next layer of LEE. In this work, we make the transformed features have a normal distribution. By doing so, we force the LEE model of the higher layer to set more $A'_{j,i}$ to nonzero values (so that the Central Limit Theorem takes effect to make $X'_i$ a Gaussian distribution), which leads to more global structures at the higher layer. We used two methods to find this activation function in our experiments.

**Parametric Activation Function**

Assume $s$ approximately follows a generalized Gaussian distribution(GGD). The probability density function of a GGD is given by:

$$f(s; \sigma, \theta) = \frac{\theta}{2\sigma\Gamma(1/\theta)} \exp\{-\left|\frac{s}{\sigma}\right|^\theta\} \tag{4}$$

where $\sigma > 0$ is a scale parameter and $\theta > 0$ is a shape parameter and $\Gamma$ denotes the gamma function. These two parameters can be estimated efficiently by an iterative algorithm developed by [17].

$s$ is then transformed into a normally distributed $N(0,1)$ random variable by the function $g$:

$$u = g(|s|) = F^{-1}\Big(\frac{\gamma(\frac{|s|^\theta}{\sigma^\theta}, \frac{1}{\theta})}{\Gamma(\frac{1}{\theta})}\Big) \tag{5}$$

where $F$ denotes the cumulative density function (cdf) of standard normal distribution and $\gamma$ denotes the incomplete gamma function. This transformation can be seen as three consecutive steps:

- Discard the sign: $u \leftarrow |s|$, now $u$ bears pdf $g(u; \sigma, \theta) = \frac{\theta}{\sigma\Gamma(\frac{1}{\theta})} \exp\{-\frac{u}{\sigma}^\theta\}$, $0 \leq u < \infty$ and cdf $\frac{\gamma(\frac{|s|^\theta}{\sigma^\theta}, \frac{1}{\theta})}{\Gamma(\frac{1}{\theta})}$, $0 \leq u < \infty$.

- Transform to a uniform distribution $U[0,1]$ by applying its own cdf: $u \leftarrow \frac{\gamma(\frac{|s|^\theta}{\sigma^\theta}, \frac{1}{\theta})}{\Gamma(\frac{1}{\theta})}$.

- Transform to a Gaussian distribution by applying the inverse cdf of $N(0,1)$: $u \leftarrow F^{-1}(u)$.

**Nonparametric Activation Function**

When the number of samples $N$ is sufficiently large, a non-parametric activation function works more efficiently. In this approach, all the samples $|S_i|$ are sorted in ascending order. For each sample $s$, $cdf(|s|)$ is approximated by the ratio of its ranking in the list with $N$. Then $u = F^{-1}(\widehat{cdf}(|s|))$ will approximately follow the standard normal distribution. Note that since $u_i$ depends only on the rank order of $|s_i|$, the results would be the same if the signs are discarded by taking $s_i^2$.

## 4 Experiments on Natural Images

To test the behavior of our model, we applied it to small patches taken from digitized natural images. The image dataset is available on the World Wide Web from Bruno Olshausen [4]. It contains ten 512x512 pre-whitened images. We took 151,290 evenly distributed 20x20 image patches. We ran the FastICA algorithm [15] and obtained 397 basis functions. As reported in other models, the basis functions are Gabor-like filters (Figure 4).

The nonparametric method was used to transform the marginal distribution of the outputs' absolute values to a standard normal distribution. Then the FastICA algorithm was applied again to retrieve 100 basis functions[5]. We adopted the visualization method employed by [12] to investigate what kind of structures the second layer units are fond of. The basis functions are fitted to Gabor filter functions using a gradient descent algorithm [12]. The connection weights from a layer-2 unit to layer-1 units are shown in Figure 5, arranged by either the center or frequency/orientation of the fitted Gabor filters. The layer-2 units are qualitatively similar to those found in [18]. Some units welcome strong activation of layer-1 units within a certain orientation range but have no preference for locations, while others have a location preference but welcome activation of layer-1 units of all frequencies and orientations, and some develop a picky appetite for both.

Again, the nonparametric method was used to transform the marginal distribution of the absolute values of the outputs from the second layer to a standard normal distribution, and FastICA was applied to retrieve 20 basis functions. We had no initial guess of what kind of statistical structure these third layer units might capture. The activation map of a couple of these units, however, seemed to suggest that they might be tuned to respond to complicated textures. In particular, one unit seems more activated by seemingly blank background, while another seems to like textures of leaves (Figure 6). We think that probably a larger database than merely 10 images, and larger image patches would be helpful for producing cleaner high level units.

The same procedure can be repeated for multiple layers. However, at this point, until we develop better methods for analyzing the representation developed by these deeply embedded units, we will leave this for future work.

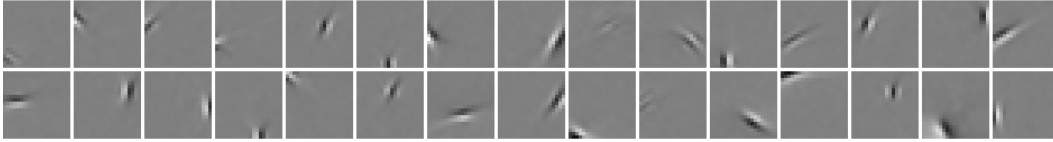

Figure 4: A subset of the 397 ICA image basis functions. Each basis function is 20x20 pixels. They are 2D Gabor like filters.

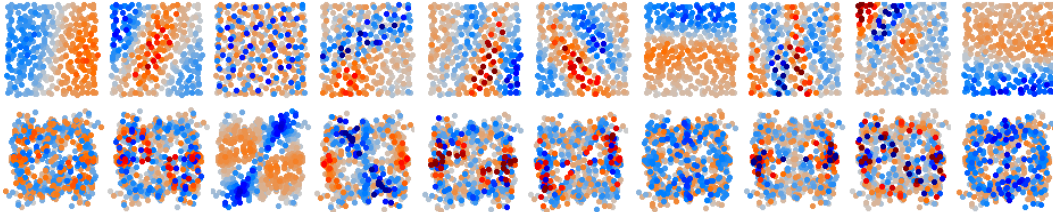

Figure 5: Sample units from the second layer. The upper panel arranges the connection weights from layer-2 units to layer-1 units by the centers of the fitted Gabor filters. Every point corresponds to one basis function of the first layer, located at the center of the fitted Gabor filter. Warm colors represent strong positive connections; cold colors represent negative connections. For example, the leftmost unit prefers strong activation of layer-1 units located on the right and weak activation of layer-1 units on the left. The lower panel arranges the connection weights by the frequencies and the orientations of the fitted Gabor filters. Now every point corresponds to the Gabor filter's frequency and orientation (in polar coordinates). The third leftmost unit welcomes strong activation of Gabor filters whose orientations are around $\frac{3}{4}\pi$ but prefers no/little activation from those whose orientations are around $\frac{1}{4}\pi$.

## 5 Discussion

The key idea of our model is to transform the high-order residual redundancy to linear dependency that can be easily exploited again by the LEE model. By using activation functions that are dependent on the marginal distribution of the outputs, a normal Gaussian interface is provided at every layer. This procedure can then repeat itself and a hierarchical model with same structure at every level can thus be constructed. As the redundancy is reduced progressively along the layers, statistical structures are also captured to progressively higher orders.

Our simulation of a three layer Recursive ICA shows the effectiveness of our model. The first layer, not surprisingly, produces the Gabor like basis functions as linear ICA always does. The second layer, however, produces basis functions that qualitatively resemble those produced by a previous hierarchical generative model [7]. This is remarkable given that our model is essentially a filtering model with no assumptions of underlying independent variables, but merely targeting redundancy reduction. The advantage of our model is the theoretical simplicity of generalization to a third layer or more. For the Karklin and Lewicki model, the assumption that the ultimate independent causal variables are two layers away from the images has to be reworked for a three layer system. It is not clear how the variables at every layer should affect the next when an extra layer is added. Osindero et al. [8] employed an energy based model. The energy function used at the first layer made it essentially a linear ICA algorithm, thus it also produces Gabor like filters. The first layer outputs are squared to discard the signs and then fed to the next layer. The inputs for the second layer are thus all positive and bear a very different marginal distribution from those for the first layer. The energy function is changed accordingly and the second layer is essentially doing nonnegative ICA. The output of this layer, however, will all be positive, which makes discarding the signs no longer an effective way of exposing higher-order dependence. Thus, to extend to another layer, new activation functions and new energy function must be derived.

The third layer of our model produces some interesting results in that some units seem to have preferences for complicated textures (Figure 6). However, as the statistical structure represented here must be of very high order, we are still looking for an effective visualization method. Also, as

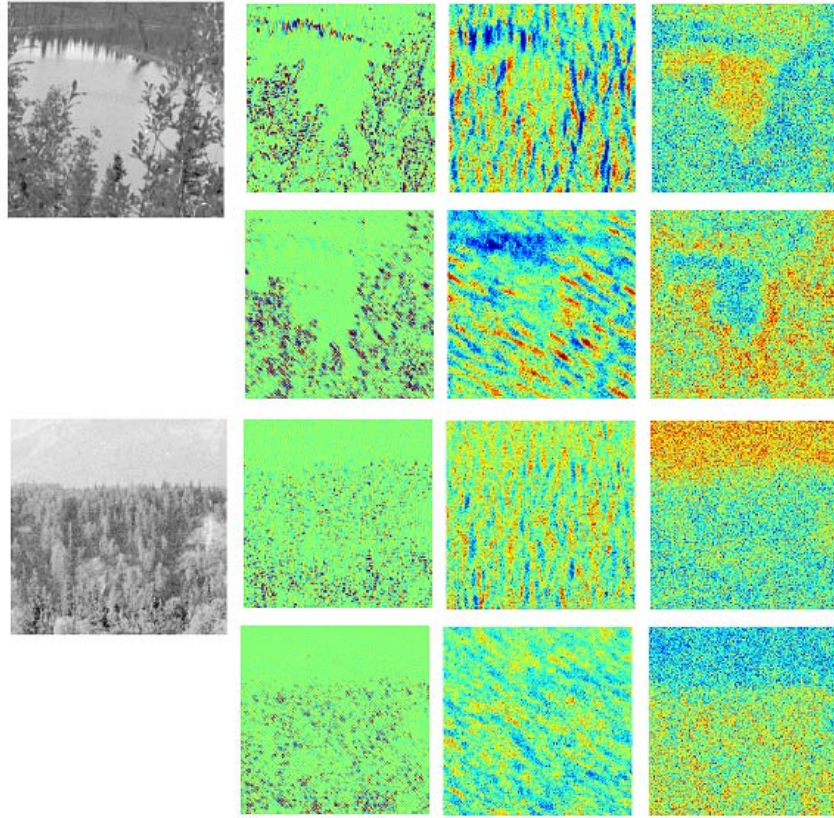

Figure 6: Activation maps on two images (upper and lower panel respectively) for two units per layer. The leftmost two images are the raw images. The second left column to the rightmost column are activation maps of two units from the first layer to the third respectively. The first layer units respond to small local edges, the second layer units respond to larger borders, and the third layer units seem to respond to large area of textures.

units at the second layer have larger receptive field than those at the first layer, it is reasonable to expect the third layer to bear even larger ones. We believe that a wider range of visual structure will be picked up by the third layer units with a larger patch size on a larger training set.

**Acknowledgments**

We thank Eric Wiewiora, Lei Zhang and members from GURU for helpful discussions. This work was supported by NIH grant MH57075 to GWC.

## Footnotes

[1]We refer to such algorithms as linear efficient encoding (LEE) algorithms throughout this paper.

[2] $p(X)$ is symmetric if $X$ and $-X$ have the same distribution.

[3] For all the image patches we tried, the distribution of pixel values on whitened image patches can be well fitted by a generalized Gaussian distribution. This is true even for small image patches. The only exception we have discovered occurs when the original image contains only binomial noise.

[4] http://redwood.berkeley.edu/bruno/sparsenet/

[5] This reduction in the number of units follows the example of [18]. In general, there appears to be less information in later layers (as assessed by eigenvalue analysis), most likely due to the discarding of the sign.

# References

[1] Anthony J. Bell and Terrence J. Sejnowski. The 'independent components' of natural scenes are edge filters. *Vision Research*, 37(23):3327–3338, 1997.

[2] Bruno A. Olshausen and David J. Field. Emergence of simple-cell receptive field properties by learning a sparse code for natural images. *Nature*, 381:607–609, 1996.

[3] Michael S. Lewicki. Efficient coding of natural sounds. *Nature Neuroscience*, 5(4):356–363, 2002.

[4] Odelia Schwartz and Eero P. Simoncelli. Natural signal statistics and sensory gain control. *Nature Neuroscience*, 4(8):819–825, 2001.

[5] Aapo Hyvarinen and Patrik O. Hoyer. A two-layer sparse coding model learns simple and complex cell receptive fields and topography from natural images. *Vision Research*, 41(18):2413–2423, 2001.

[6] Martin J. Wainwright and Eero P. Simoncelli. Scale mixtures of Gaussians and the statistics of natural images. In *Advances in Neural Information Processing Systems*, volume 12, pages 855–861, Cambridge, MA, May 2000. MIT Press.

[7] Yan Karklin and Michael S. Lewicki. A hierarchical bayesian model for learning non-linear statistical regularities in non-stationary natural signals. *Neural Computation*, 17(2):397–423, 2005.

[8] Simon Osindero, Max Welling, and Geoffrey E. Hinton. Topographic Product Models Applied to Natural Scene Statistics. *Neural Computation*, 18:381–414, 2005.

[9] Eva M. Finney, Ione Fine, and Karen R. Dobkins. Visual stimuli activate auditory cortex in the deaf. *Nature Neuroscience*, 4:1171–1173, 2001.

[10] Horace B. Barlow. Redundancy reduction revisited. *Network: Computation in Neural Systems*, 12:241–253, 2001.

[11] Horace B. Barlow. Possible principles underlying the transformation of sensory messages. In Walter A. Rosenblith, editor, *Sensory Communication*, pages 217–234. MIT Press, Cambridge, MA, USA, 1961.

[12] Michael S. Lewicki and Bruno A. Olshausen. A probabilistic framework for the adaptation and comparison of image codes. *Journal of the Optical Society of America A*, 16(7):1587–1601, 1999.

[13] Yee Whye Teh, Max Welling, Simon Osindero, and Geoffrey E. Hinton. Energy-based models for sparse overcomplete representations. *Journal of Machine Learning Research*, 4:1235–1260, 2003.

[14] David Attwell and Simon B. Laughlin. An energy budget for signaling in the grey matter of the brain. *Journal of Cerebral Blood Flow and Metabolism*, 21(10):1133–1145, 2001.

[15] Aapo Hyvarinen and Erkki Oja. A fast fixed-point algorithm for independent component analysis. *Neural Computation*, 9(7):1483–1492, 1997.

[16] David J. Field. What is the goal of sensory coding? *Neural Computation*, 6(4):559–601, 1994.

[17] Kai-Sheng Song. A globally convergent and consistent method for estimating the shape parameter of a generalized Gaussian distribution. *IEEE Transactions on Information Theory*, 52(2):510–527, 2006.

[18] Yan Karklin and Michael S. Lewicki. Learning higher-order structures in natural images. *Network: Computation in Neural Systems*, 14:483–499, 2003.
